# Active Learning with a Drifting Distribution

**Liu Yang**
Machine Learning Department
Carnegie Mellon University
liuy@cs.cmu.edu

## Abstract

We study the problem of active learning in a stream-based setting, allowing the distribution of the examples to change over time. We prove upper bounds on the number of prediction mistakes and number of label requests for established disagreement-based active learning algorithms, both in the realizable case and under Tsybakov noise. We further prove minimax lower bounds for this problem.

## 1 Introduction

Most existing analyses of active learning are based on an i.i.d. assumption on the data. In this work, we assume the data are independent, but we allow the distribution from which the data are drawn to shift over time, while the target concept remains fixed. We consider this problem in a stream-based selective sampling model, and are interested in two quantities: the number of mistakes the algorithm makes on the first $T$ examples in the stream, and the number of label requests among the first $T$ examples in the stream.

In particular, we study scenarios in which the distribution may drift within a fixed totally bounded family of distributions. Unlike previous models of distribution drift [Bar92, CMEDV10], the minimax number of mistakes (or excess number of mistakes, in the noisy case) can be *sublinear* in the number of samples.

We specifically study the classic CAL active learning strategy [CAL94] in this context, and bound the number of mistakes and label requests the algorithm makes in the realizable case, under conditions on the concept space and the family of possible distributions. We also exhibit lower bounds on these quantities that match our upper bounds in certain cases. We further study a noise-robust variant of CAL, and analyze its number of mistakes and number of label requests in noisy scenarios where the noise distribution remains fixed over time but the marginal distribution on $\mathcal{X}$ may shift. In particular, we upper bound these quantities under Tsybakov's noise conditions [MT99]. We also prove minimax lower bounds under these same conditions, though there is a gap between our upper and lower bounds.

## 2 Definition and Notations

As in the usual statistical learning problem, there is a standard Borel space $\mathcal{X}$, called the instance space, and a set $\mathbb{C}$ of measurable classifiers $h : \mathcal{X} \to \{-1, +1\}$, called the concept space. We additionally have a space $\mathbb{D}$ of distributions on $\mathcal{X}$, called the distribution space. Throughout, we suppose that the VC dimension of $\mathbb{C}$, denoted $d$ below, is finite.

For any $\mu_1, \mu_2 \in \mathbb{D}$, let $\|\mu_1 - \mu_2\| = \sup_A \mu_1(A) - \mu_2(A)$ denote the total variation pseudo-distance between $\mu_1$ and $\mu_2$, where the set $A$ in the sup ranges over all measurable subsets of $\mathcal{X}$. For any $\epsilon > 0$, let $\mathbb{D}_\epsilon$ denote a minimal $\epsilon$-cover of $\mathbb{D}$, meaning that $\mathbb{D}_\epsilon \subseteq \mathbb{D}$ and $\forall \mu_1 \in \mathbb{D}, \exists \mu_2 \in \mathbb{D}_\epsilon$ s.t. $\|\mu_1 - \mu_2\| < \epsilon$, and that $\mathbb{D}_\epsilon$ has minimal possible size $|\mathbb{D}_\epsilon|$ among all subsets of $\mathbb{D}$ with this property.

In the learning problem, there is an unobservable sequence of distributions $\mathcal{D}_1, \mathcal{D}_2, \ldots$, with each $\mathcal{D}_t \in \mathbb{D}$, and an unobservable time-independent regular conditional distribution, which we represent

by a function $\eta : \mathcal{X} \to [0, 1]$. Based on these quantities, we let $\mathcal{Z} = \{(X_t, Y_t)\}_{t=1}^{\infty}$ denote an infinite sequence of independent random variables, such that $\forall t, X_t \sim \mathcal{D}_t$, and the conditional distribution of $Y_t$ given $X_t$ satisfies $\forall x \in \mathcal{X}, \mathbb{P}(Y_t = +1 | X_t = x) = \eta(x)$. Thus, the joint distribution of $(X_t, Y_t)$ is specified by the pair $(\mathcal{D}_t, \eta)$, and the distribution of $\mathcal{Z}$ is specified by the collection $\{\mathcal{D}_t\}_{t=1}^{\infty}$ along with $\eta$. We also denote by $\mathcal{Z}_t = \{(X_1, Y_1), (X_2, Y_2), \dots, (X_t, Y_t)\}$ the first $t$ such labeled examples. Note that the $\eta$ conditional distribution is time-independent, since we are restricting ourselves to discussing drifting marginal distributions on $\mathcal{X}$, rather than drifting concepts. Concept drift is an important and interesting topic, but is beyond the scope of our present discussion.

In the active learning protocol, at each time $t$, the algorithm is presented with the value $X_t$, and is required to predict a label $\hat{Y}_t \in \{-1, +1\}$; then after making this prediction, it may optionally request to observe the true label value $Y_t$; as a means of book-keeping, if the algorithm requests a label $Y_t$ on round $t$, we define $Q_t = 1$, and otherwise $Q_t = 0$.

We are primarily interested in two quantities. The first, $\hat{M}_T = \sum_{t=1}^{T} \mathbb{I}\left[\hat{Y}_t \neq Y_t\right]$, is the cumulative number of mistakes up to time $T$. The second quantity of interest, $\hat{Q}_T = \sum_{t=1}^{T} Q_t$, is the total number of labels requested up to time $T$. In particular, we will study the expectations of these quantities: $\bar{M}_T = \mathbb{E}\left[\hat{M}_T\right]$ and $\bar{Q}_T = \mathbb{E}\left[\hat{Q}_T\right]$. We are particularly interested in the asymptotic dependence of $\bar{Q}_T$ and $\bar{M}_T - \bar{M}_T^*$ on $T$, where $\bar{M}_T^* = \inf_{h \in \mathbb{C}} \mathbb{E}\left[\sum_{t=1}^{T} \mathbb{I}\left[h(X_t) \neq Y_t\right]\right]$. We refer to $\bar{Q}_T$ as the expected number of label requests, and to $\bar{M}_T - \bar{M}_T^*$ as the expected excess number of mistakes. For any distribution $P$ on $\mathcal{X}$, we define $\mathrm{er}_P(h) = \mathbb{E}_{X \sim P}[\eta(X)\mathbb{I}[h(X) = -1] + (1 - \eta(X))\mathbb{I}[h(X) = +1]]$, the probability of $h$ making a mistake for $X \sim P$ and $Y$ with conditional probability of being $+1$ equal $\eta(X)$. Note that, abbreviating $\mathrm{er}_t(h) = \mathrm{er}_{\mathcal{D}_t}(h) = \mathbb{P}(h(X_t) \neq Y_t)$, we have $\bar{M}_T^* = \inf_{h \in \mathbb{C}} \sum_{t=1}^{T} \mathrm{er}_t(h)$.

Scenarios in which both $\bar{M}_T - \bar{M}_T^*$ and $\bar{Q}_T$ are $o(T)$ (i.e., sublinear) are considered desirable, as these represent cases in which we do "learn" the proper way to predict labels, while asymptotically using far fewer labels than passive learning. Once establishing conditions under which this is possible, we may then further explore the trade-off between these two quantities.

We will additionally make use of the following notions. For $V \subseteq \mathbb{C}$, let $\mathrm{diam}_t(V) = \sup_{h, g \in V} \mathcal{D}_t(\{x : h(x) \neq g(x)\})$. For $h : \mathcal{X} \to \{-1, +1\}$, $\bar{\mathrm{er}}_{s:t}(h) = \frac{1}{t-s+1} \sum_{u=s}^{t} \mathrm{er}_u(h)$, and for finite $S \subseteq \mathcal{X} \times \{-1, +1\}$, $\hat{\mathrm{er}}(h; S) = \frac{1}{|S|} \sum_{(x,y) \in S} \mathbb{I}[h(x) \neq y]$. Also let $\mathbb{C}[S] = \{h \in \mathbb{C} : \hat{\mathrm{er}}(h; S) = 0\}$. Finally, for a distribution $P$ on $\mathcal{X}$ and $r > 0$, define $\mathrm{B}_P(h, r) = \{g \in \mathbb{C} : P(x : h(x) \neq g(x)) \leq r\}$.

## 2.1 Assumptions

In addition to the assumption of independence of the $X_t$ variables and that $d < \infty$, each result below is stated under various additional assumptions. The weakest such assumption is that $\mathbb{D}$ is *totally bounded*, in the following sense. For each $\epsilon > 0$, let $\mathbb{D}_\epsilon$ denote a minimal subset of $\mathbb{D}$ such that $\forall \mathcal{D} \in \mathbb{D}, \exists \mathcal{D}' \in \mathbb{D}_\epsilon$ s.t. $\|\mathcal{D} - \mathcal{D}'\| < \epsilon$: that is, a minimal $\epsilon$-cover of $\mathbb{D}$. We say that $\mathbb{D}$ is totally bounded if it satisfies the following assumption.

*Assumption* 1. $\forall \epsilon > 0, |\mathbb{D}_\epsilon| < \infty$.

In some of the results below, we will be interested in deriving specific rates of convergence. Doing so requires us to make stronger assumptions about $\mathbb{D}$ than mere total boundedness. We will specifically consider the following condition, in which $c, m \in [0, \infty)$ are constants.

*Assumption* 2. $\forall \epsilon > 0, |\mathbb{D}_\epsilon| < c \cdot \epsilon^{-m}$.

For an example of a class $\mathbb{D}$ satisfying the total boundedness assumption, consider $\mathcal{X} = [0, 1]^n$, and let $\mathbb{D}$ be the collection of distributions that have uniformly continuous density function with respect to the Lebesgue measure on $\mathcal{X}$, with modulus of continuity at most some value $\omega(\epsilon)$ for each value of $\epsilon > 0$, where $\omega(\epsilon)$ is a fixed real-valued function with $\lim_{\epsilon \to 0} \omega(\epsilon) = 0$.

As a more concrete example, when $\omega(\epsilon) = L\epsilon$ for some $L \in (0, \infty)$, this corresponds to the family of Lipschitz continuous density functions with Lipschitz constant at most $L$. In this case, we have $|\mathbb{D}_\epsilon| \leq O(\epsilon^{-n})$, satisfying Assumption 2.

# 3 Related Work

We discuss active learning under distribution drift, with fixed target concept. There are several branches of the literature that are highly relevant to this, including domain adaptation [MMR09, MMR08], online learning [Lit88], learning with concept drift, and empirical processes for independent but not identically distributed data [vdG00].

**Streamed-based Active Learning with a Fixed Distribution**  [DKM09] show that a certain modified perceptron-like active learning algorithm can achieve a mistake bound $O(d \log(T))$ and query bound $\tilde{O}(d \log(T))$, when learning a linear separator under a uniform distribution on the unit sphere, in the realizable case. [DGS10] also analyze the problem of learning linear separators under a uniform distribution, but allowing Tsybakov noise. They find that with $\bar{Q}_T = \tilde{O}\left(d^{\frac{2\alpha}{\alpha+2}} T^{\frac{2}{\alpha+2}}\right)$ queries, it is possible to achieve an expected excess number of mistakes $\bar{M}_T - M_T^* = \tilde{O}\left(d^{\frac{\alpha+1}{\alpha+2}} \cdot T^{\frac{1}{\alpha+2}}\right)$. At this time, we know of no work studying the number of mistakes and queries achievable by active learning in a stream-based setting where the distribution may change over time.

**Stream-based Passive Learning with a Drifting Distribution**  There has been work on learning with a drifting distribution and fixed target, in the context of passive learning. [Bar92, BL97] study the problem of learning a subset of a domain from randomly chosen examples when the probability distribution of the examples changes slowly but continually throughout the learning process; they give upper and lower bounds on the best achievable probability of misclassification after a given number of examples. They consider learning problems in which a changing environment is modeled by a slowly changing distribution on the product space. The allowable drift is restricted by ensuring that consecutive probability distributions are close in total variation distance. However, this assumption allows for certain malicious choices of distribution sequences, which shift the probability mass into smaller and smaller regions where the algorithm is uncertain of the target's behavior, so that the number of mistakes grows linearly in the number of samples in the worst case. More recently, [FM97] have investigated learning when the distribution changes as a linear function of time. They present algorithms that estimate the error of functions, using knowledge of this linear drift.

# 4 Active Learning in the Realizable Case

Throughout this section, suppose $\mathbb{C}$ is a fixed concept space and $h^* \in \mathbb{C}$ is a fixed target function: that is, $\mathrm{er}_t(h^*) = 0$. The family of scenarios in which this is true are often collectively referred to as the *realizable case*. We begin our analysis by studying this realizable case because it greatly simplifies the analysis, laying bare the core ideas in plain form. We will discuss more general scenarios, in which $\mathrm{er}_t(h^*) \geq 0$, in later sections, where we find that essentially the same principles apply there as in this initial realizable-case analysis.

We will be particularly interested in the performance of the following simple algorithm, due to [CAL94], typically referred to as CAL after its discoverers. The version presented here is specified in terms of a passive learning subroutine $\mathcal{A}$ (mapping any sequence of labeled examples to a classifier). In it, we use the notation $\mathrm{DIS}(V) = \{x \in \mathcal{X} : \exists h, g \in V \text{ s.t. } h(x) \neq g(x)\}$, also used below.

---

CAL
1. $t \leftarrow 0$, $\mathcal{Q}_0 \leftarrow \emptyset$, and let $\hat{h}_0 = \mathcal{A}(\emptyset)$
2. Do
3.    $t \leftarrow t + 1$
4.    Predict $\hat{Y}_t = \hat{h}_{t-1}(X_t)$
5.    If $\max\limits_{y \in \{-1,+1\}} \min\limits_{h \in \mathbb{C}} \hat{\mathrm{er}}(h; \mathcal{Q}_{t-1} \cup \{(X_t, y)\}) = 0$
6.      Request $Y_t$, let $\mathcal{Q}_t = \mathcal{Q}_{t-1} \cup \{(X_t, Y_t)\}$
7.    Else let $Y_t' = \operatorname*{argmin}\limits_{y \in \{-1,+1\}} \min\limits_{h \in \mathbb{C}} \hat{\mathrm{er}}(h; \mathcal{Q}_{t-1} \cup \{(X_t, y)\})$, and let $\mathcal{Q}_t \leftarrow \mathcal{Q}_{t-1} \cup \{(X_t, Y_t')\}$
8.    Let $\hat{h}_t = \mathcal{A}(\mathcal{Q}_t)$

---

Below, we let $\mathcal{A}_{1IG}$ denote the one-inclusion graph prediction strategy of [HLW94]. Specifically, the passive learning algorithm $\mathcal{A}_{1IG}$ is specified as follows. For a sequence of data points $\mathcal{U} \in \mathcal{X}^{t+1}$,

the one-inclusion graph is a graph, where each vertex represents a distinct labeling of $\mathcal{U}$ that can be realized by some classifier in $\mathbb{C}$, and two vertices are adjacent if and only if their corresponding labelings for $\mathcal{U}$ differ by exactly one label. We use the one-inclusion graph to define a classifier based on $t$ training points as follows. Given $t$ labeled data points $\mathcal{L} = \{(x_1, y_1), \ldots, (x_t, y_t)\}$, and one test point $x_{t+1}$ we are asked to predict a label for, we first construct the one-inclusion graph on $\mathcal{U} = \{x_1, \ldots, x_{t+1}\}$; we then orient the graph (give each edge a unique direction) in a way that minimizes the maximum out-degree, and breaks ties in a way that is invariant to permutations of the order of points in $\mathcal{U}$; after orienting the graph in this way, we examine the subset of vertices whose corresponding labeling of $\mathcal{U}$ is consistent with $\mathcal{L}$; if there is only one such vertex, then we predict for $x_{t+1}$ the corresponding label from that vertex; otherwise, if there are two such vertices, then they are adjacent in the one-inclusion graph, and we choose the one toward which the edge is directed and use the label for $x_{t+1}$ in the corresponding labeling of $\mathcal{U}$ as our prediction for the label of $x_{t+1}$. See [HLW94] and subsequent work for detailed studies of the one-inclusion graph prediction strategy.

### 4.1 Learning with a Fixed Distribution
We begin the discussion with the simplest case: namely, when $|\mathbb{D}| = 1$.

**Definition 1.** *[Han07, Han11] Define the disagreement coefficient of $h^*$ under a distribution $P$ as*

$$\theta_P(\epsilon) = \sup_{r > \epsilon} P\left(\text{DIS}(\text{B}_P(h^*, r))\right)/r.$$

**Theorem 1.** *For any distribution $P$ on $\mathcal{X}$, if $\mathbb{D} = \{P\}$, then running CAL with $\mathcal{A} = \mathcal{A}_{1IG}$ achieves expected mistake bound $\bar{M}_T = O\left(d\log(T)\right)$ and expected query bound $\bar{Q}_T = O\left(\theta_P(\epsilon_T)d\log^2(T)\right)$, for $\epsilon_T = d\log(T)/T$.*

For completeness, the proof is included in the supplemental materials.

### 4.2 Learning with a Drifting Distribution
We now generalize the above results to any sequence of distributions from a totally bounded space $\mathbb{D}$. Throughout this section, let $\theta_{\mathbb{D}}(\epsilon) = \sup_{P \in \mathbb{D}} \theta_P(\epsilon)$.

First, we prove a basic result stating that CAL can achieve a sublinear number of mistakes, and under conditions on the disagreement coefficient, also a sublinear number of queries.

**Theorem 2.** *If $\mathbb{D}$ is totally bounded (Assumption 1), then CAL (with $\mathcal{A}$ any empirical risk minimization algorithm) achieves an expected mistake bound $\bar{M}_T = o(T)$, and if $\theta_{\mathbb{D}}(\epsilon) = o(1/\epsilon)$, then CAL makes an expected number of queries $\bar{Q}_T = o(T)$.*

*Proof.* As mentioned, given that $\text{er}_{\mathcal{Q}_{t-1}}(h^*) = 0$, we have that $Y_t'$ in Step 7 must equal $h^*(X_t)$, so that the invariant $\text{er}_{\mathcal{Q}_t}(h^*) = 0$ is maintained for all $t$ by induction. In particular, this implies $\mathcal{Q}_t = \mathcal{Z}_t$ for all $t$.

Fix any $\epsilon > 0$, and enumerate the elements of $\mathbb{D}_\epsilon$ so that $\mathbb{D}_\epsilon = \{P_1, P_2, \ldots, P_{|\mathbb{D}_\epsilon|}\}$. For each $t \in \mathbb{N}$, let $k(t) = \text{argmin}_{k \leq |\mathbb{D}_\epsilon|} \|P_k - \mathcal{D}_t\|$, breaking ties arbitrarily. Let

$$L(\epsilon) = \left\lceil \frac{8}{\sqrt{\epsilon}} \left( d \ln\left(\frac{24}{\sqrt{\epsilon}}\right) + \ln\left(\frac{4}{\sqrt{\epsilon}}\right) \right) \right\rceil.$$

For each $i \leq |\mathbb{D}_\epsilon|$, if $k(t) = i$ for infinitely many $t \in \mathbb{N}$, then let $T_i$ denote the smallest value of $T$ such that $|\{t \leq T : k(t) = i\}| = L(\epsilon)$. If $k(t) = i$ only finitely many times, then let $T_i$ denote the largest index $t$ for which $k(t) = i$, or $T_i = 1$ if no such index $t$ exists.

Let $T_\epsilon = \max_{i \leq |\mathbb{D}_\epsilon|} T_i$ and $V_\epsilon = \mathbb{C}[\mathcal{Z}_{T_\epsilon}]$. We have that $\forall t > T_\epsilon, \text{diam}_t(V_\epsilon) \leq \text{diam}_{k(t)}(V_\epsilon) + \epsilon$. For each $i$, let $\mathcal{L}_i$ be a sequence of $L(\epsilon)$ i.i.d. pairs $(X, Y)$ with $X \sim P_i$ and $Y = h^*(X)$, and let $V_i = \mathbb{C}[\mathcal{L}_i]$. Then $\forall t > T_\epsilon$,

$$\mathbb{E}\left[\text{diam}_{k(t)}(V_\epsilon)\right] \leq \mathbb{E}\left[\text{diam}_{k(t)}(V_{k(t)})\right] + \sum_{s \leq T_i : k(s) = k(t)} \|\mathcal{D}_s - P_{k(s)}\| \leq \mathbb{E}\left[\text{diam}_{k(t)}(V_{k(t)})\right] + L(\epsilon)\epsilon.$$

By classic results in the theory of PAC learning [AB99, Vap82] and our choice of $L(\epsilon)$, $\forall t > T_\epsilon, \mathbb{E}\left[\text{diam}_{k(t)}(V_{k(t)})\right] \leq \sqrt{\epsilon}$.

Combining the above arguments,

$$\mathbb{E}\left[\sum_{t=1}^{T} \operatorname{diam}_t(\mathbb{C}[\mathcal{Z}_{t-1}])\right] \leq T_\epsilon + \sum_{t=T_\epsilon+1}^{T} \mathbb{E}\left[\operatorname{diam}_t(V_\epsilon)\right] \leq T_\epsilon + \epsilon T + \sum_{t=T_\epsilon+1}^{T} \mathbb{E}\left[\operatorname{diam}_{k(t)}(V_\epsilon)\right]$$

$$\leq T_\epsilon + \epsilon T + L(\epsilon)\epsilon T + \sum_{t=T_\epsilon+1}^{T} \mathbb{E}\left[\operatorname{diam}_{k(t)}(V_{k(t)})\right]$$

$$\leq T_\epsilon + \epsilon T + L(\epsilon)\epsilon T + \sqrt{\epsilon}T.$$

Let $\epsilon_T$ be any nonincreasing sequence in $(0,1)$ such that $1 \ll T_{\epsilon_T} \ll T$. Since $|\mathbb{D}_\epsilon| < \infty$ for all $\epsilon > 0$, we must have $\epsilon_T \to 0$. Thus, noting that $\lim_{\epsilon \to 0} L(\epsilon)\epsilon = 0$, we have

$$\mathbb{E}\left[\sum_{t=1}^{T} \operatorname{diam}_t(\mathbb{C}[\mathcal{Z}_{t-1}])\right] \leq T_{\epsilon_T} + \epsilon_T T + L(\epsilon_T)\epsilon_T T + \sqrt{\epsilon_T}T \ll T. \qquad (1)$$

The result on $\bar{M}_T$ now follows by noting that for any $\hat{h}_{t-1} \in \mathbb{C}[\mathcal{Z}_{t-1}]$ has $\operatorname{er}_t(\hat{h}_{t-1}) \leq \operatorname{diam}_t(\mathbb{C}[\mathcal{Z}_{t-1}])$, so

$$\bar{M}_T = \mathbb{E}\left[\sum_{t=1}^{T} \operatorname{er}_t\left(\hat{h}_{t-1}\right)\right] \leq \mathbb{E}\left[\sum_{t=1}^{T} \operatorname{diam}_t(\mathbb{C}[\mathcal{Z}_{t-1}])\right] \ll T.$$

Similarly, for $r > 0$, we have

$$\mathbb{P}(\text{Request } Y_t) = \mathbb{E}\left[\mathbb{P}(X_t \in \operatorname{DIS}(\mathbb{C}[\mathcal{Z}_{t-1}])|\mathcal{Z}_{t-1})\right] \leq \mathbb{E}\left[\mathbb{P}(X_t \in \operatorname{DIS}(\mathbb{C}[\mathcal{Z}_{t-1}] \cup \operatorname{B}_{\mathcal{D}_t}(h^*, r)))\right]$$
$$\leq \mathbb{E}\left[\theta_{\mathbb{D}}(r) \cdot \max\{\operatorname{diam}_t(\mathbb{C}[\mathcal{Z}_{t-1}]), r\}\right] \leq \theta_{\mathbb{D}}(r) \cdot r + \theta_{\mathbb{D}}(r) \cdot \mathbb{E}\left[\operatorname{diam}_t(\mathbb{C}[\mathcal{Z}_{t-1}])\right].$$

Letting $r_T = T^{-1}\mathbb{E}\left[\sum_{t=1}^{T} \operatorname{diam}_t(\mathbb{C}[\mathcal{Z}_{t-1}])\right]$, we see that $r_T \to 0$ by (1), and since $\theta_{\mathbb{D}}(\epsilon) = o(1/\epsilon)$, we also have $\theta_{\mathbb{D}}(r_T)r_T \to 0$, so that $\theta_{\mathbb{D}}(r_T)r_T T \ll T$. Therefore, $\bar{Q}_T$ equals

$$\sum_{t=1}^{T} \mathbb{P}(\text{Request } Y_t) \leq \theta_{\mathbb{D}}(r_T) \cdot r_T \cdot T + \theta_{\mathbb{D}}(r_T) \cdot \mathbb{E}\left[\sum_{t=1}^{T} \operatorname{diam}_t(\mathbb{C}[\mathcal{Z}_{t-1}])\right] = 2\theta_{\mathbb{D}}(r_T) \cdot r_T \cdot T \ll T. \qquad \square$$

We can also state a more specific result in the case when we have some more detailed information on the sizes of the finite covers of $\mathbb{D}$.

**Theorem 3.** *If Assumption 2 is satisfied, then CAL (with $\mathcal{A}$ any empirical risk minimization algorithm) achieves an expected mistake bound $\bar{M}_T$ and expected number of queries $\bar{Q}_T$ such that $\bar{M}_T = O\left(T^{\frac{m}{m+1}} d^{\frac{1}{m+1}} \log^2 T\right)$ and $\bar{Q}_T = O\left(\theta_{\mathbb{D}}(\epsilon_T) T^{\frac{m}{m+1}} d^{\frac{1}{m+1}} \log^2 T\right)$, where $\epsilon_T = (d/T)^{\frac{1}{m+1}}$.*

*Proof.* Fix $\epsilon > 0$, enumerate $\mathbb{D}_\epsilon = \{P_1, P_2, \ldots, P_{|\mathbb{D}_\epsilon|}\}$, and for each $t \in \mathbb{N}$, let $k(t) = \operatorname{argmin}_{1 \leq k \leq |\mathbb{D}_\epsilon|} \|\mathcal{D}_t - P_k\|$. Let $\{X'_t\}_{t=1}^{\infty}$ be a sequence of independent samples, with $X'_t \sim P_{k(t)}$, and $\mathcal{Z}'_t = \{(X'_1, h^*(X'_1)), \ldots, (X'_t, h^*(X'_t))\}$. Then

$$\mathbb{E}\left[\sum_{t=1}^{T} \operatorname{diam}_t(\mathbb{C}[\mathcal{Z}_{t-1}])\right] \leq \mathbb{E}\left[\sum_{t=1}^{T} \operatorname{diam}_t(\mathbb{C}[\mathcal{Z}'_{t-1}])\right] + \sum_{t=1}^{T} \|\mathcal{D}_t - P_{k(t)}\|$$

$$\leq \mathbb{E}\left[\sum_{t=1}^{T} \operatorname{diam}_t(\mathbb{C}[\mathcal{Z}'_{t-1}])\right] + \epsilon T \leq \sum_{t=1}^{T} \mathbb{E}\left[\operatorname{diam}_{P_{k(t)}}(\mathbb{C}[\mathcal{Z}'_{t-1}])\right] + 2\epsilon T.$$

The classic convergence rates results from PAC learning [AB99, Vap82] imply

$$\sum_{t=1}^{T} \mathbb{E}\left[\operatorname{diam}_{P_{k(t)}}(\mathbb{C}[\mathcal{Z}'_{t-1}])\right] = \sum_{t=1}^{T} O\left(\frac{d \log t}{|\{i \leq t : k(i) = k(t)\}|}\right)$$

$$\leq O(d \log T) \cdot \sum_{t=1}^{T} \frac{1}{|\{i \leq t : k(i) = k(t)\}|} \leq O(d \log T) \cdot |\mathbb{D}_\epsilon| \cdot \sum_{u=1}^{\lceil T/|\mathbb{D}_\epsilon| \rceil} \frac{1}{u} \leq O\left(d|\mathbb{D}_\epsilon| \log^2(T)\right).$$

Thus, $\sum_{t=1}^{T} \mathbb{E}\left[\operatorname{diam}_t(\mathbb{C}[\mathcal{Z}_{t-1}])\right] \leq O\left(d|\mathbb{D}_\epsilon| \log^2(T) + \epsilon T\right) \leq O\left(d \cdot \epsilon^{-m} \log^2(T) + \epsilon T\right)$.
Taking $\epsilon = (T/d)^{-\frac{1}{m+1}}$, this is $O\left(d^{\frac{1}{m+1}} \cdot T^{\frac{m}{m+1}} \log^2(T)\right)$. We therefore have

$$\bar{M}_T \leq \mathbb{E}\left[\sum_{t=1}^{T} \sup_{h \in \mathbb{C}[\mathcal{Z}_{t-1}]} \operatorname{er}_t(h)\right] \leq \mathbb{E}\left[\sum_{t=1}^{T} \operatorname{diam}_t(\mathbb{C}[\mathcal{Z}_{t-1}])\right] \leq O\left(d^{\frac{1}{m+1}} \cdot T^{\frac{m}{m+1}} \log^2(T)\right).$$

Similarly, letting $\epsilon_T = (d/T)^{\frac{1}{m+1}}$, $\bar{Q}_T$ is at most

$$\mathbb{E}\left[\sum_{t=1}^{T} \mathcal{D}_t(\operatorname{DIS}(\mathbb{C}[\mathcal{Z}_{t-1}]))\right] \leq \mathbb{E}\left[\sum_{t=1}^{T} \mathcal{D}_t\left(\operatorname{DIS}\left(\operatorname{B}_{\mathcal{D}_t}\left(h^*, \max\left\{\operatorname{diam}_t(\mathbb{C}[\mathcal{Z}_{t-1}]), \epsilon_T\right\}\right)\right)\right)\right]$$

$$\leq \mathbb{E}\left[\sum_{t=1}^{T} \theta_{\mathbb{D}}\left(\epsilon_T\right) \cdot \max\left\{\operatorname{diam}_t(\mathbb{C}[\mathcal{Z}_{t-1}]), \epsilon_T\right\}\right]$$

$$\leq \mathbb{E}\left[\sum_{t=1}^{T} \theta_{\mathbb{D}}\left(\epsilon_T\right) \cdot \operatorname{diam}_t(\mathbb{C}[\mathcal{Z}_{t-1}])\right] + \theta_{\mathbb{D}}\left(\epsilon_T\right) T \epsilon_T \leq O\left(\theta_{\mathbb{D}}\left(\epsilon_T\right) \cdot d^{\frac{1}{m+1}} \cdot T^{\frac{m}{m+1}} \log^2(T)\right). \ \square$$

We can additionally construct a lower bound for this scenario, as follows. Suppose $\mathbb{C}$ contains a full infinite binary tree for which all classifiers in the tree agree on some point. That is, there is a set of points $\{x_b : b \in \{0,1\}^k, k \in \mathbb{N}\}$ such that, for $b_1 = 0$ and $\forall b_2, b_3, \ldots \in \{0,1\}$, $\exists h \in \mathbb{C}$ such that $h(x_{(b_1, \ldots, b_{j-1})}) = b_j$ for $j \geq 2$. For instance, this is the case for linear separators (and most other natural "geometric" concept spaces).

**Theorem 4.** *For any $\mathbb{C}$ as above, for any active learning algorithm, $\exists$ a set $\mathbb{D}$ satsifying Assumption 2, a target function $h^* \in \mathbb{C}$, and a sequence of distributions $\{\mathcal{D}_t\}_{t=1}^{T}$ in $\mathbb{D}$ such that the achieved $\bar{M}_T$ and $\bar{Q}_T$ satisfy $\bar{M}_T = \Omega\left(T^{\frac{m}{m+1}}\right)$, and $\bar{M}_T = O\left(T^{\frac{m}{m+1}}\right) \implies \bar{Q}_T = \Omega\left(T^{\frac{m}{m+1}}\right)$.*

The proof is analogous to that of Theorem 9 below, and is therefore omitted for brevity.

## 5  Learning with Noise

In this section, we extend the above analysis to allow for various types of noise conditions commonly studied in the literature. For this, we will need to study a noise-robust variant of CAL, below referred to as Agnostic CAL (or ACAL). We prove upper bounds achieved by ACAL, as well as (non-matching) minimax lower bounds.

### 5.1  Noise Conditions

The following assumption may be referred to as a *strictly benign noise* condition, which essentially says the model is specified correctly in that $h^* \in \mathbb{C}$, and though the labels may be stochastic, they are not completely random, but rather each is slightly biased toward the $h^*$ label.

*Assumption* 3. $h^* = \operatorname{sign}(\eta - 1/2) \in \mathbb{C}$ and $\forall x, \eta(x) \neq 1/2$.

A particularly interesting special case of Assumption 3 is given by Tsybakov's noise conditions, which essentially control how common it is to have $\eta$ values close to $1/2$. Formally:

*Assumption* 4. $\eta$ satisfies Assumption 3 and for some $c > 0$ and $\alpha \geq 0$,
$\forall t > 0, P(|\eta(x) - 1/2| < t) < c \cdot t^\alpha$.

In the setting of shifting distributions, we will be interested in conditions for which the above assumptions are satisifed simultaneously for all distributions in $\mathbb{D}$. We formalize this in the following.

*Assumption* 5. Assumption 4 is satisfied for all $\mathcal{D} \in \mathbb{D}$, with the same $c$ and $\alpha$ values.

### 5.2  Agnostic CAL

The following algorithm is essentially taken from [DHM07, Han11], adapted here for this stream-based setting. It is based on a subroutine: $\operatorname{LEARN}(\mathcal{L}, \mathcal{Q}) = \operatorname*{argmin}_{h \in \mathbb{C}: \hat{\operatorname{er}}(h; \mathcal{L})=0} \hat{\operatorname{er}}(h; \mathcal{Q})$ if $\min_{h \in \mathbb{C}} \hat{\operatorname{er}}(h; \mathcal{L}) = 0$, and otherwise $\operatorname{LEARN}(\mathcal{L}, \mathcal{Q}) = \varnothing$.

```
ACAL
1.  t ← 0, 𝓛_t ← ∅, 𝓠_t ← ∅, let ĥ_t be any element of ℂ
2.  Do
3.    t ← t + 1
4.    Predict Ŷ_t = ĥ_{t−1}(X_t)
5.    For each y ∈ {−1, +1}, let h^{(y)} = LEARN(𝓛_{t−1}, 𝓠_{t−1})
6.    If either y has h^{(−y)} = ∅ or
              êr(h^{(−y)}; 𝓛_{t−1} ∪ 𝓠_{t−1}) − êr(h^{(y)}; 𝓛_{t−1} ∪ 𝓠_{t−1}) > Ê_{t−1}(𝓛_{t−1}, 𝓠_{t−1})
7.      𝓛_t ← 𝓛_{t−1} ∪ {(X_t, y)}, 𝓠_t ← 𝓠_{t−1}
8.    Else Request Y_t, and let 𝓛_t ← 𝓛_{t−1}, 𝓠_t ← 𝓠_{t−1} ∪ {(X_t, Y_t)}
9.    Let ĥ_t = LEARN(𝓛_t, 𝓠_t)
10.   If t is a power of 2
11.     𝓛_t ← ∅, 𝓠_t ← ∅
```

The algorithm is expressed in terms of a function $\hat{\mathcal{E}}_t(\mathcal{L}, \mathcal{Q})$, defined as follows. Let $\delta_i$ be a nonincreasing sequence of values in $(0,1)$. Let $\xi_1, \xi_2, \ldots$ denote a sequence of independent Uniform$(\{-1, +1\})$ random variables, also independent from the data. For $V \subseteq \mathbb{C}$, let $\hat{R}_t(V) = \sup_{h_1, h_2 \in V} \frac{1}{t - 2^{\lfloor \log_2(t-1) \rfloor}} \sum_{m = 2^{\lfloor \log_2(t-1) \rfloor} + 1}^{t} \xi_m \cdot (h_1(X_m) - h_2(X_m))$, $\hat{D}_t(V) = \sup_{h_1, h_2 \in V} \frac{1}{t - 2^{\lfloor \log_2(t-1) \rfloor}} \sum_{m = 2^{\lfloor \log_2(t-1) \rfloor} + 1}^{t} |h_1(X_m) - h_2(X_m)|$, $\hat{U}_t(V, \delta) = 12\hat{R}_t(V) + 34\sqrt{\hat{D}_t(V) \frac{\ln(32t^2/\delta)}{t}} + \frac{752 \ln(32t^2/\delta)}{t}$. Also, for any finite sets $\mathcal{L}, \mathcal{Q} \subseteq \mathcal{X} \times \mathcal{Y}$, let $\mathbb{C}[\mathcal{L}] = \{h \in \mathbb{C} : \hat{er}(h; \mathcal{L}) = 0\}$, $\hat{\mathbb{C}}(\epsilon; \mathcal{L}, \mathcal{Q}) = \{h \in \mathbb{C}[\mathcal{L}] : \hat{er}(h; \mathcal{L} \cup \mathcal{Q}) - \min_{g \in \mathbb{C}[\mathcal{L}]} \hat{er}(g; \mathcal{L} \cup \mathcal{Q}) \leq \epsilon\}$. Then define $\hat{U}_t(\epsilon, \delta; \mathcal{L}, \mathcal{Q}) = \hat{U}_t(\hat{\mathbb{C}}_t(\epsilon; \mathcal{L}, \mathcal{Q}), \delta)$, and (letting $\mathbb{Z}_\epsilon = \{j \in \mathbb{Z} : 2^j \geq \epsilon\}$)

$$\hat{\mathcal{E}}_t(\mathcal{L}, \mathcal{Q}) = \inf \left\{ \epsilon > 0 : \forall j \in \mathbb{Z}_\epsilon, \min_{m \in \mathbb{N}} \hat{U}_t(\epsilon, \delta_{\lfloor \log(t) \rfloor}; \mathcal{L}, \mathcal{Q}) \leq 2^{j-4} \right\}.$$

### 5.3 Learning with a Fixed Distribution

The following results essentially follow from [Han11], adapted to this stream-based setting.

**Theorem 5.** *For any strictly benign $(P, \eta)$, if $2^{-2^i} \ll \delta_i \ll 2^{-i}/i$, ACAL achieves an expected excess number of mistakes $\bar{M}_T - M_T^* = o(T)$, and if $\theta_P(\epsilon) = o(1/\epsilon)$, then ACAL makes an expected number of queries $\bar{Q}_T = o(T)$.*

**Theorem 6.** *For any $(P, \eta)$ satisfying Assumption 4, if $\mathbb{D} = \{P\}$, ACAL achieves an expected excess number of mistakes $\bar{M}_T - M_T^* = \tilde{O}\left(d^{\frac{1}{\alpha+2}} \cdot T^{\frac{\alpha+1}{\alpha+2}} \log\left(\frac{1}{\delta_{\lfloor \log(T) \rfloor}}\right) + \sum_{i=0}^{\lfloor \log(T) \rfloor} \delta_i 2^i\right)$, and an expected number of queries $\bar{Q}_T = \tilde{O}\left(\theta_P(\epsilon_T) \cdot d^{\frac{2}{\alpha+2}} \cdot T^{\frac{\alpha}{\alpha+2}} \log\left(\frac{1}{\delta_{\lfloor \log(T) \rfloor}}\right) + \sum_{i=0}^{\lfloor \log(T) \rfloor} \delta_i 2^i\right)$. where $\epsilon_T = T^{-\frac{\alpha}{\alpha+2}}$.*

**Corollary 1.** *For any $(P, \eta)$ satisfying Assumption 4, if $\mathbb{D} = \{P\}$ and $\delta_i = 2^{-i}$ in ACAL, the algorithm achieves an expected number of mistakes $\bar{M}_T$ and expected number of queries $\bar{Q}_T$ such that, for $\epsilon_T = T^{-\frac{\alpha}{\alpha+2}}$, $\bar{M}_T - M_T^* = \tilde{O}\left(d^{\frac{1}{\alpha+2}} \cdot T^{\frac{\alpha+1}{\alpha+2}}\right)$, and $\bar{Q}_T = \tilde{O}\left(\theta_P(\epsilon_T) \cdot d^{\frac{2}{\alpha+2}} \cdot T^{\frac{\alpha}{\alpha+2}}\right)$.*

### 5.4 Learning with a Drifting Distribution

We can now state our results concerning ACAL, which are analogous to Theorems 2 and 3 proved earlier for CAL in the realizable case.

**Theorem 7.** *If $\mathbb{D}$ is totally bounded (Assumption 1) and $\eta$ satisfies Assumption 3, then ACAL with $\delta_i = 2^{-i}$ achieves an excess expected mistake bound $\bar{M}_T - M_T^* = o(T)$, and if additionally $\theta_{\mathbb{D}}(\epsilon) = o(1/\epsilon)$, then ACAL makes an expected number of queries $\bar{Q}_T = o(T)$.*

The proof of Theorem 7 essentially follows from a combination of the reasoning for Theorem 2 and Theorem 8 below. Its proof is omitted.

**Theorem 8.** *If Assumptions 2 and 5 are satisfied, then ACAL achieves an expected excess number of mistakes $\bar{M}_T - M_T^* = \tilde{O}\left(T^{\frac{(\alpha+2)m+1}{(\alpha+2)(m+1)}} \log\left(\frac{1}{\delta_{\lfloor \log(T) \rfloor}}\right) + \sum_{i=0}^{\lfloor \log(T) \rfloor} \delta_i 2^i\right)$, and an expected number of queries $\bar{Q}_T = \tilde{O}\left(\theta_{\mathbb{D}}(\epsilon_T) T^{\frac{(\alpha+2)(m+1)-\alpha}{(\alpha+2)(m+1)}} \log\left(\frac{1}{\delta_{\lfloor \log(T) \rfloor}}\right) + \sum_{i=0}^{\lfloor \log(T) \rfloor} \delta_i 2^i\right)$, where $\epsilon_T = T^{-\frac{\alpha}{(\alpha+2)(m+1)}}$.*

The proof of this result is in many ways similar to that given above for the realizable case, and is included among the supplemental materials.

We immediately have the following corollary for a specific $\delta_i$ sequence.

**Corollary 2.** *With $\delta_i = 2^{-i}$ in ACAL, the algorithm achieves expected number of mistakes $\bar{M}$ and expected number of queries $\bar{Q}_T$ such that, for $\epsilon_T = T^{-\frac{\alpha}{(\alpha+2)(m+1)}}$,*

$$\bar{M}_T - M_T^* = \tilde{O}\left(T^{\frac{(\alpha+2)m+1}{(\alpha+2)(m+1)}}\right) \text{ and } \bar{Q}_T = \tilde{O}\left(\theta_{\mathbb{D}}(\epsilon_T) \cdot T^{\frac{(\alpha+2)(m+1)-\alpha}{(\alpha+2)(m+1)}}\right).$$

Just as in the realizable case, we can also state a minimax lower bound for this noisy setting.

**Theorem 9.** *For any $\mathbb{C}$ as in Theorem 4, for any active learning algorithm, $\exists$ a set $\mathbb{D}$ satisfying Assumption 2, a conditional distribution $\eta$, such that Assumption 5 is satisfied, and a sequence of distributions $\{\mathcal{D}_t\}_{t=1}^T$ in $\mathbb{D}$ such that the $\bar{M}_T$ and $\bar{Q}_T$ achieved by the learning algorithm satisfy $\bar{M}_T - M_T^* = \Omega\left(T^{\frac{1+m\alpha}{\alpha+2+m\alpha}}\right)$ and $\bar{M}_T - M_T^* = O\left(T^{\frac{1+m\alpha}{\alpha+2+m\alpha}}\right) \implies \bar{Q}_T = \Omega\left(T^{\frac{2+m\alpha}{\alpha+2+m\alpha}}\right).$*

The proof is included in the supplemental material.

## 6 Discussion

**Querying before Predicting:** One interesting alternative to the above framework is to allow the learner to make a label request *before* making its label predictions. From a practical perspective, this may be more desirable and in many cases quite realistic. From a theoretical perspective, analysis of this alternative framework essentially separates out the mistakes due to over-confidence from the mistakes due to recognized uncertainty. In some sense, this is related to the KWIK model of learning of [LLW08].

Analyzing the above procedures in this alternative model yields several interesting details. Specifically, the natural modification of CAL produces a method that (in the realizable case) makes the same number of label requests as before, except that now it makes *zero* mistakes, since CAL will request a label if there is *any* uncertainty about its label.

On the other hand, the analysis of the natural modification to ACAL can be far more subtle, when there is noise. In particular, because the version space is only guaranteed to contain the best classifier *with high confidence*, there is still a small probability of making a prediction that disagrees with the best classifier $h^*$ on each round that we do not request a label. So controlling the number of mistakes in this setting comes down to controlling the probability of removing $h^*$ from the version space. However, this confidence parameter appears in the analysis of the number of queries, so that we have a natural trade-off between the number of mistakes and the number of label requests. In particular, under Assumptions 2 and 5, this procedure achieves an expected excess number of mistakes $\bar{M}_T - M_T^* \leq \sum_{i=1}^{\lfloor \log(T)\rfloor} \delta_i 2^i$, and an expected number of queries $\bar{Q}_T = \tilde{O}\left(\theta_{\mathbb{D}}(\epsilon_T) \cdot T^{\frac{(\alpha+2)(m+1)-\alpha}{(\alpha+2)(m+1)}} \log\left(\frac{1}{\delta_{\lfloor \log(T)\rfloor}}\right) + \sum_{i=0}^{\lfloor \log(T)\rfloor} \delta_i 2^i\right)$, where $\epsilon_T = T^{-\frac{\alpha}{(\alpha+2)(m+1)}}$. In particular, given any nondecreasing sequence $M_T$, we can set this $\delta_i$ sequence to maintain $\bar{M}_T - M_T^* \leq M_T$ for all $T$.

**Open Problems:** What is not implied by the results above is any sort of *trade-off* between the number of mistakes and the number of queries. Intuitively, such a trade-off should exist; however, as CAL lacks any parameter to adjust the behavior with respect to this trade-off, it seems we need a different approach to address that question. In the batch setting, the analogous question is the trade-off between the number of label requests and the number of unlabeled examples needed. In the realizable case, that trade-off is tightly characterized by Dasgupta's *splitting index* analysis [Das05]. It would be interesting to determine whether the splitting index tightly characterizes the mistakes-vs-queries trade-off in this stream-based setting as well.

In the batch setting, in which unlabeled examples are considered free, and performance is only measured as a function of the number of label requests, [BHV10] have found that there is an important distinction between the *verifiable* label complexity and the *unverifiable* label complexity. In particular, while the former is sometimes no better than passive learning, the latter can always provide improvements for VC classes. Is there such a thing as unverifiable performance measures in the stream-based setting? To be concrete, we have the following open problem. Is there a method for every VC class that achieves $O(\log(T))$ mistakes and $o(T)$ queries in the realizable case?

# References

[AB99]      M. Anthony and P. L. Bartlett. *Neural Network Learning: Theoretical Foundations*. Cambridge University Press, 1999.

[Bar92]     P. L. Bartlett. Learning with a slowly changing distribution. In *Proceedings of the fifth annual workshop on Computational learning theory*, COLT '92, pages 243–252, 1992.

[BHV10]    M.-F. Balcan, S. Hanneke, and J. Wortman Vaughan. The true sample complexity of active learning. *Machine Learning*, 80(2–3):111–139, September 2010.

[BL97]      R. D. Barve and P. M. Long. On the complexity of learning from drifting distributions. *Inf. Comput.*, 138(2):170–193, 1997.

[CAL94]     D. Cohn, L. Atlas, and R. Ladner. Improving generalization with active learning. *Machine Learning*, 15(2):201–221, 1994.

[CMEDV10] K. Crammer, Y. Mansour, E. Even-Dar, and J. Wortman Vaughan. Regret minimization with concept drift. In *COLT*, pages 168–180, 2010.

[Das05]      S. Dasgupta. Coarse sample complexity bounds for active learning. In *Advances in Neural Information Processing Systems 18*, 2005.

[DGS10]     O. Dekel, C. Gentile, and K. Sridharam. Robust selective sampling from single and multiple teachers. In *Conference on Learning Theory*, 2010.

[DHM07]    S. Dasgupta, D. Hsu, and C. Monteleoni. A general agnostic active learning algorithm. Technical Report CS2007-0898, Department of Computer Science and Engineering, University of California, San Diego, 2007.

[DKM09]    S. Dasgupta, A. Kalai, and C. Monteleoni. Analysis of perceptron-based active learning. *Journal of Machine Learning Research*, 10:281–299, 2009.

[FM97]      Y. Freund and Y. Mansour. Learning under persistent drift. In *Proceedings of the Third European Conference on Computational Learning Theory*, EuroCOLT '97, pages 109–118, 1997.

[Han07]     S. Hanneke. A bound on the label complexity of agnostic active learning. In *Proceedings of the $24^{th}$ International Conference on Machine Learning*, 2007.

[Han11]     S. Hanneke. Rates of convergence in active learning. *The Annals of Statistics*, 39(1):333–361, 2011.

[HLW94]    D. Haussler, N. Littlestone, and M. Warmuth. Predicting $\{0, 1\}$-functions on randomly drawn points. *Information and Computation*, 115:248–292, 1994.

[Lit88]       N. Littlestone. Learning quickly when irrelevant attributes abound: A new linear-threshold algorithm. *Machine Learning*, 2:285–318, 1988.

[LLW08]     L. Li, M. L. Littman, and T. J. Walsh. Knows what it knows: A framework for self-aware learning. In *International Conference on Machine Learning*, 2008.

[MMR08]    Y. Mansour, M. Mohri, and A. Rostamizadeh. Domain adaptation with multiple sources. In *In Advances in Neural Information Processing Systems (NIPS)*, pages 1041–1048, 2008.

[MMR09]    Y. Mansour, M. Mohri, and A. Rostamizadeh. Domain adaptation: Learning bounds and algorithms. In *COLT*, 2009.

[MT99]      E. Mammen and A.B. Tsybakov. Smooth discrimination analysis. *The Annals of Statistics*, 27:1808–1829, 1999.

[Vap82]      V. Vapnik. *Estimation of Dependencies Based on Empirical Data*. Springer-Verlag, New York, 1982.

[vdG00]     S. van de Geer. *Empirical Processes in M-Estimation (Cambridge Series in Statistical and Probabilistic Mathematics)*. Cambridge University Press, 2000.

